# Dual Estimation and the Unscented Transformation

Eric A. Wan
*ericwan@ece.ogi.edu*

Rudolph van der Merwe
*rudmerwe@ece.ogi.edu*

Alex T. Nelson
*atnelson@ece.ogi.edu*

Oregon Graduate Institute of Science & Technology
Department of Electrical and Computer Engineering
20000 N.W. Walker Rd., Beaverton, Oregon 97006

## Abstract

Dual estimation refers to the problem of simultaneously estimating the state of a dynamic system and the model which gives rise to the dynamics. Algorithms include expectation-maximization (EM), dual Kalman filtering, and joint Kalman methods. These methods have recently been explored in the context of *nonlinear* modeling, where a neural network is used as the functional form of the unknown model. Typically, an extended Kalman filter (EKF) or smoother is used for the part of the algorithm that estimates the clean state given the current estimated model. An EKF may also be used to estimate the weights of the network. This paper points out the flaws in using the EKF, and proposes an improvement based on a new approach called the *unscented transformation* (UT) [3]. A substantial performance gain is achieved with the same order of computational complexity as that of the standard EKF. The approach is illustrated on several dual estimation methods.

## 1  Introduction

We consider the problem of learning both the hidden states $x_k$ and parameters $w$ of a discrete-time nonlinear dynamic system,

$$x_{k+1} = F(x_k, v_k, w) \qquad (1)$$
$$y_k = H(x_k, n_k, w), \qquad (2)$$

where $y_k$ is the only observed signal. The *process* noise $v_k$ drives the dynamic system, and the *observation* noise is given by $n_k$. Note that we are not assuming additivity of the noise sources.

A number of approaches have been proposed for this problem. The *dual EKF* algorithm uses two separate EKFs: one for signal estimation, and one for model estimation. The states are estimated given the current weights and the weights are estimated given the current states. In the *joint EKF*, the state and model parameters are concatenated within a combined state vector, and a single EKF is used to estimate both quantities simultaneously. The *EM* algorithm uses an extended Kalman smoother for the E-step, in which forward and

backward passes are made through the data to estimate the signal. The model is updated during a separate M-step.

For a more thorough treatment and a theoretical basis on how these algorithms relate, see Nelson [6]. Rather than provide a comprehensive comparison between the different algorithms, the goal of this paper is to point out the assumptions and flaws in the EKF (Section 2), and offer a improvement based on the unscented transformation/filter (Section 3). The *unscented filter* has recently been proposed as a substitute for the EKF in nonlinear control problems (known dynamic model) [3]. This paper presents new research on the use of the UF within the dual estimation framework for both state and weight estimation. In the case of weight estimation, the UF represents a new efficient "second-order" method for training neural networks in general.

## 2   Flaws in the EKF

Assume for now that we know the model (weight parameters) for the dynamic system in Equations 1 and 2. Given the noisy observation $\mathbf{y}_k$, a recursive estimation for $\hat{\mathbf{x}}_k$ can be expressed in the form,

$$\hat{\mathbf{x}}_k = (\text{optimal prediction of } \mathbf{x}_k) + G_k \times [\mathbf{y}_k - (\text{optimal prediction of } \mathbf{y}_k)] \qquad (3)$$

This recursion provides the optimal MMSE estimate for $\mathbf{x}_k$ assuming the prior estimate $\hat{\mathbf{x}}_k$ and current observation $\mathbf{y}_k$ are Gaussian. We need not assume linearity of the model. The optimal terms in this recursion are given by

$$\hat{\mathbf{x}}_k^- = E[F(\hat{\mathbf{x}}_{k-1}, \mathbf{v}_{k-1})] \qquad G_k = \mathbf{P}_{\mathbf{x}_k \mathbf{y}_k} \mathbf{P}_{\tilde{\mathbf{y}}_k \tilde{\mathbf{y}}_k}^{-1} \qquad \hat{\mathbf{y}}_k^- = E[H(\hat{\mathbf{x}}_k^-, \mathbf{n}_k)], \qquad (4)$$

where the optimal prediction $\hat{\mathbf{x}}_k^-$ is the expectation of a nonlinear function of the random variables $\hat{\mathbf{x}}_{k-1}$ and $\mathbf{v}_{k-1}$ (similar interpretation for the optimal prediction of $\mathbf{y}_k$). The optimal gain term is expressed as a function of posterior covariance matrices (with $\tilde{\mathbf{y}}_k = \mathbf{y}_k - \hat{\mathbf{y}}_k^-$). Note these terms also require taking expectations of a nonlinear function of the prior state estimates.

The Kalman filter calculates these quantities exactly in the linear case. For nonlinear models, however, the extended KF approximates these as:

$$\hat{\mathbf{x}}_k^- \approx F(\hat{\mathbf{x}}_{k-1}, \bar{\mathbf{v}}) \qquad G_k \approx \hat{\mathbf{P}}_{\mathbf{x}_k \mathbf{y}_k} \hat{\mathbf{P}}_{\tilde{\mathbf{y}}_k \tilde{\mathbf{y}}_k}^{-1} \qquad \hat{\mathbf{y}}_k^- = H(\hat{\mathbf{x}}_k^-, \bar{\mathbf{n}}), \qquad (5)$$

where predictions are approximated as simply the function of the prior *mean* value for estimates (no expectation taken). The covariance are determined by linearizing the dynamic equations ($\mathbf{x}_{k+1} \approx A\mathbf{x}_k + B\mathbf{v}_k$, $\mathbf{y}_k \approx C\mathbf{x}_k + D\mathbf{n}_k$), and then determining the posterior covariance matrices analytically for the linear system. As such, the EKF can be viewed as providing "first-order" approximations to the optimal terms (in the sense that expressions are approximated using a first-order Taylor series expansion of the nonlinear terms around the mean values). While "second-order" versions of the EKF exist, their increased implementation and computational complexity tend to prohibit their use.

## 3   The Unscented Transformation/Filter

The unscented transformation (UT) is a method for calculating the statistics of a random variable which undergoes a nonlinear transformation [3]. Consider propagating a random variable $\alpha$ (dimension $L$) through a nonlinear function, $\beta = g(\alpha)$. Assume $\alpha$ has mean $\bar{\alpha}$ and covariance $\mathbf{P}_\alpha$. To calculate the statistics of $\beta$, we form a matrix $\mathcal{X}$ of $2L + 1$ *sigma* vectors $\mathcal{X}_i$, where the first vector ($\mathcal{X}_0$) corresponds to $\bar{\alpha}$, and the rest are computed from the mean (+)plus and (-)minus each column of the matrix square-root of $\mathbf{P}_\alpha$. These sigma

vectors are propagated through the nonlinear function, and the mean and covariance for $\beta$ are approximated using a weighted sample mean and covariance,

$$\bar{\beta} \approx \frac{1}{L+\kappa} \left\{ \kappa g(\mathcal{X}_0) + \frac{1}{2} \sum_{i=1}^{2L} g(\mathcal{X}_i) \right\}, \qquad (6)$$

$$\mathbf{P}_\beta \approx \frac{1}{L+\kappa} \left\{ \kappa [g(\mathcal{X}_0) - \bar{\beta}][g(\mathcal{X}_0) - \bar{\beta}]^T + \frac{1}{2} \sum_{i=1}^{2L} [g(\mathcal{X}_i) - \bar{\beta}][g(\mathcal{X}_i) - \bar{\beta}]^T \right\} \quad (7)$$

where $\kappa$ is a scaling factor. Note that this method differs substantially from general "sampling" methods (e.g., Monte-Carlo methods and particle filters [1]) which require orders of magnitude more sample points in an attempt to propagate an accurate (possibly non-Gaussian) distribution of the state. The UT approximations are accurate to the third order for Gaussian inputs for all nonlinearities. For non-Gaussian inputs, approximations are accurate to at least the second-order, with the accuracy determined by the choice of $\kappa$ [3]. A simple example is shown in Figure 1 for a 2-dimensional system: the left plots shows the true mean and covariance propagation using Monte-Carlo sampling; the center plots show the performance of the UT (note only 5 sigma points are required); the right plots show the results using a linearization approach as would be done in the EKF. The superior performance of the UT is clear.

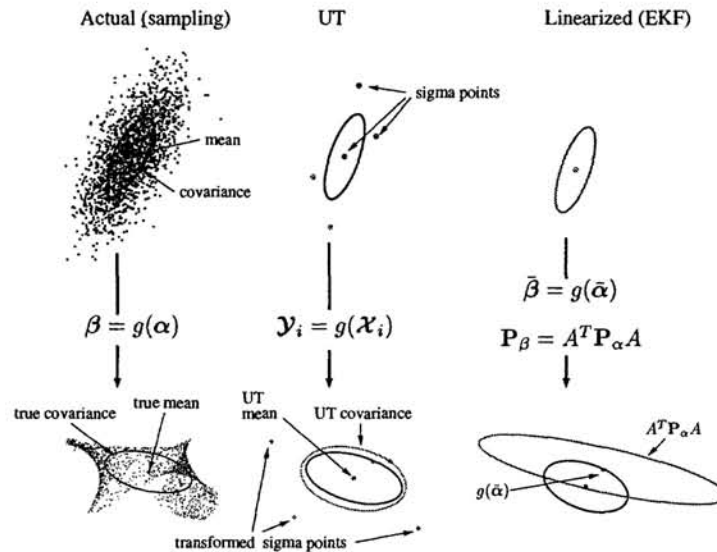

Figure 1: Example of the UT for mean and covariance propagation. a) actual, b) UT, c) first-order linear (EKF).

The *unscented filter* (UF) [3] is a straightforward extension of the UT to the recursive estimation in Equation 3, where we set $\alpha = \hat{x}_k$, and denote the corresponding sigma matrix as $\mathcal{X}(k|k)$. The UF equations are given on the next page. It is interesting to note that no explicit calculation of Jacobians or Hessians are necessary to implement this algorithm. The total number of computations is only order $L^2$ as compared to $L^3$ for the EKF.[1]

## 4  Application to Dual Estimation

This section shows the use of the UF within several dual estimation approaches. As an application domain for comparison, we consider modeling a noisy time-series as a nonlinear

UF Equations

$$W_0 = \kappa/(L + \kappa) \quad , \quad W_1 \dots W_{2L} = 1/2(L + \kappa)$$

$$\mathcal{X}(k|k-1) = \mathbf{F}[\mathcal{X}(k-1|k-1), \mathbf{P}_{vv}^{1/2}]$$

$$\hat{\mathbf{x}}_k^- = \sum_{i=0}^{2L} W_i \mathcal{X}_i(k|k-1)$$

$$\mathbf{P}_k^- = \sum_{i=0}^{2L} W_i [\mathcal{X}_i(k|k-1) - \hat{\mathbf{x}}_k^-][\mathcal{X}_i(k|k-1) - \hat{\mathbf{x}}_k^-]^T$$

$$\mathcal{Y}(k|k-1) = \mathbf{H}[\mathcal{X}(k|k-1), \mathbf{P}_{nn}^{1/2}]$$

$$\hat{\mathbf{y}}_k^- = \sum_{i=0}^{2L} W_i \mathcal{Y}_i(k|k-1)$$

$$\mathbf{P}_{\tilde{\mathbf{y}}_k \tilde{\mathbf{y}}_k} = \sum_{i=0}^{2L} W_i [\mathcal{Y}_i(k|k-1) - \hat{\mathbf{y}}_k^-][\mathcal{Y}_i(k|k-1) - \hat{\mathbf{y}}_k^-]^T$$

$$\mathbf{P}_{\mathbf{x}_k \mathbf{y}_k} = \sum_{i=0}^{2L} W_i [\mathcal{X}_i(k|k-1) - \hat{\mathbf{x}}_k^-][\mathcal{Y}_i(k|k-1) - \hat{\mathbf{y}}_k^-]^T$$

$$\hat{\mathbf{x}}_k = \hat{\mathbf{x}}_k^- + \mathbf{P}_{\mathbf{x}_k \mathbf{y}_k} \mathbf{P}_{\tilde{\mathbf{y}}_k \tilde{\mathbf{y}}_k}^{-1} (\mathbf{y}_k - \hat{\mathbf{y}}_k^-)$$

$$\mathbf{P}_k = \mathbf{P}_k^- - \mathbf{P}_{\mathbf{x}_k \mathbf{y}_k} (\mathbf{P}_{\tilde{\mathbf{y}}_k \tilde{\mathbf{y}}_k}^{-1})^T \mathbf{P}_{\mathbf{x}_k \mathbf{y}_k}^T$$

autoregression:

$$x_k = f\left(x_{k-1}, \dots x_{k-M}, \mathbf{w}\right) + v_k$$
$$y_k = x_k + n_k, \qquad \forall k \in \{1 \dots N\} \tag{8}$$

The underlying clean signal $x_k$ is a nonlinear function of its past $M$ values, driven by white Gaussian process noise $v_k$ with variance $\sigma_v^2$. The observed data point $y_k$ includes the additive noise $n_k$, which is assumed to be Gaussian with variance $\sigma_n^2$. The corresponding state-space representation for the signal $x_k$ is given by:

$$\mathbf{x}_k = F(\mathbf{x}_{k-1}, \mathbf{w}) + B \cdot v_{k-1} \tag{9}$$

$$\begin{bmatrix} x_k \\ x_{k-1} \\ \vdots \\ x_{k-M+1} \end{bmatrix} = \begin{bmatrix} f(x_{k-1}, \dots, x_{k-M}, \mathbf{w}) \\ \begin{bmatrix} 1 & 0 & 0 & 0 \\ 0 & \ddots & 0 & \vdots \\ 0 & 0 & 1 & 0 \end{bmatrix} \cdot \begin{bmatrix} x_{k-1} \\ \vdots \\ x_{k-M} \end{bmatrix} \end{bmatrix} + \begin{bmatrix} 1 \\ 0 \\ \vdots \\ 0 \end{bmatrix} \cdot v_{k-1}$$

$$y_k = [1 \quad 0 \quad \cdots \quad 0] \cdot \mathbf{x}_k + n_k \tag{10}$$

In this context, the dual estimation problem consists of simultaneously estimating the clean signal $x_k$ and the model parameters $\mathbf{w}$ from the noisy data $y_k$.

## 4.1 Dual EKF / Dual UF

One dual estimation approach is the *dual extended Kalman filter* developed in [8, 6]. The dual EKF requires separate state-space representation for the signal and the weights. A state-space representation for the weights is generated by considering them to be a stationary process with an identity state transition matrix, driven by process noise $\mathbf{u}_k$:

$$\mathbf{w}_k = \mathbf{w}_{k-1} + \mathbf{u}_k \tag{11}$$
$$y_k = f(\mathbf{x}_{k-1}, \mathbf{w}_k) + v_k + n_k. \tag{12}$$

The noisy measurement $y_k$ has been rewritten as an observation on $\mathbf{w}$. This allows the use of an EKF for weight estimation (representing a "second-order" optimization procedure) [7]. Two EKFs can now be run simultaneously for signal and weight estimation. At every time-step, the current estimate of the weights is used in the signal-filter, and the current estimate of the signal-state is used in the weight-filter.

The *dual UF/EKF* algorithm is formed by simply replacing the EKF for state-estimation with the UF while still using an EKF for weight-estimation. In the *dual UF* algorithm both state- and weight-estimation are done with the UF. Note that the state-transition is linear in the weight filter, so the nonlinearity is restricted to the measurement equation. Here, the UF gives a more exact measurement-update phase of estimation. The use of the UF for weight estimation in general is discussed in further detail in Section 5.

## 4.2  Joint EKF / Joint UF

An alternative approach to dual estimation is provided by the *joint extended Kalman filter* [4, 5]. In this framework the signal-state and weight vector are concatenated into a single, *joint* state vector: $\mathbf{z}_k = [\mathbf{x}_k^T \ \mathbf{w}_k^T]^T$. The estimation of $\mathbf{z}_k$ can be done recursively by writing the state-space equations for the joint state as:

$$\mathbf{z}_k = \begin{bmatrix} F(\mathbf{x}_{k-1}, \mathbf{w}_{k-1}) \\ \mathbf{I} \cdot \mathbf{w}_{k-1} \end{bmatrix} + \begin{bmatrix} B \cdot v_k \\ \mathbf{u}_k \end{bmatrix} \quad \text{and} \quad y_k = [1 \quad 0 \quad \cdots \quad 0] \mathbf{z}_k + n_k, \quad (13)$$

and running an EKF on the joint state-space to produce simultaneous estimates of the states $\mathbf{x}_k$ and $\mathbf{w}$. As discussed in [6], the joint EKF provides approximate MAP estimates by maximizing the joint density of the signal and weights given the noisy data. Again, our approach in this paper is to use the UF instead of the EKF to provide more accurate estimation of the state, resulting in the *joint UF* algorithm.

## 4.3  EM - Unscented Smoothing

A somewhat different iterative approach to dual estimation is given by the expectation-maximization (EM) algorithm applied to nonlinear dynamic systems [2]. In each iteration, the conditional expectation of the signal is computed, given the data and the current estimate of the model (E-step). Then the model is found that maximizes a function of this conditional mean (M-step). For linear models, the M-step can be solved in closed form. The E-step is computed with a Kalman smoother, which combines the forward-time estimated mean and covariance $(\hat{\mathbf{x}}_k^f, \mathbf{P}_k^f)$ of the signal given *past* data, with the backward-time predicted mean and covariance $(\hat{\mathbf{x}}_k^b, \mathbf{P}_k^b)$ given the *future* data, producing the following smoothed statistics given *all* the data:

$$(\mathbf{P}_k^s)^{-1} = (\mathbf{P}_k^f)^{-1} + (\mathbf{P}_k^b)^{-1} \quad (14)$$

$$\hat{\mathbf{x}}_k^s = \mathbf{P}_k^s[(\mathbf{P}_k^b)^{-1}\hat{\mathbf{x}}_k^b - (\mathbf{P}_k^f)^{-1}\hat{\mathbf{x}}_k^f]. \quad (15)$$

When a MLP neural network model is used, the M-step can no longer be computed in closed-form, and a gradient-based approach is used instead. The resulting algorithm is usually referred to as generalized EM (GEM) [2]. The E-step is typically approximated by an extended Kalman smoother, wherein a linearization of the model is used for backward propagation of the state estimates.

We propose improving the E-step of the EM algorithm for nonlinear models by using a UF instead of an EKF to compute both the forward and backward passes in the Kalman smoother. Rather than linearize the model for the backward pass, as in [2], a neural network is trained on the backward dynamics (as well as the forward dynamics). This allows for a more exact backward estimation phase using the UF, and enables the development of an *unscented smoother* (US).

### 4.4 Experiments

We present results on two simple time-series to provide a clear illustration of the use of the UF over the EKF. The first series is the Mackey-Glass chaotic series with additive WGN (SNR $\approx 3dB$). The second time series (also chaotic) comes from an autoregressive neural network with random weights driven by Gaussian process noise and also corrupted by additive WGN (SNR $\approx 3dB$). A standard 5-3-1 MLP with *tanh* hidden activation functions and a linear output layer was used in all the filters. The process and measurement noise variances were assumed to be known.

Results on training and testing data, as well as training curves for the different dual estimation methods are shown below. The quoted numbers are normalized (clean signal variance) mean-square estimation and prediction errors. The superior performance of the UT based algorithms (especially the *dual UF*) are clear. Note also the more stable learning curves using the UF approaches. These improvements have been found to be consistent and statistically significant on a number of additional experiments.

| Mackey-Glass | Train | | Test | |
|---|---|---|---|---|
| **Algorithm** | **Est.** | **Pred.** | **Est.** | **Pred.** |
| Dual EKF | 0.20 | 0.50 | 0.21 | 0.54 |
| Dual UF/EKF | 0.19 | 0.50 | 0.19 | 0.53 |
| Dual UF | 0.15 | 0.45 | 0.14 | 0.48 |
| Joint EKF | 0.22 | 0.53 | 0.22 | 0.56 |
| Joint UF | 0.19 | 0.50 | 0.18 | 0.53 |

| Chaotic AR-NN | Train | | Test | |
|---|---|---|---|---|
| **Algorithm** | **Est.** | **Pred.** | **Est.** | **Pred.** |
| Dual EKF | 0.32 | 0.62 | 0.36 | 0.69 |
| Dual UF/EKF | 0.26 | 0.58 | 0.28 | 0.69 |
| Dual UF | 0.23 | 0.55 | 0.27 | 0.63 |
| Joint EKF | 0.29 | 0.58 | 0.34 | 0.72 |
| Joint UF | 0.25 | 0.55 | 0.30 | 0.67 |

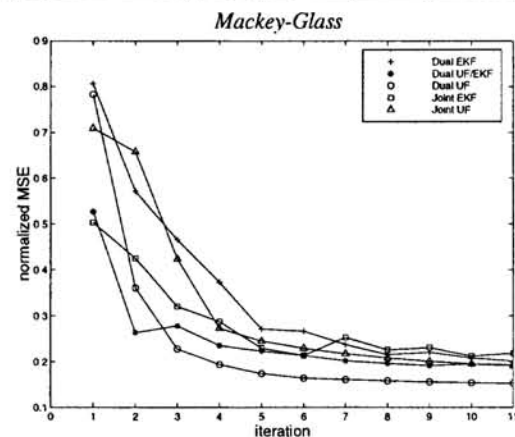

*Mackey-Glass*

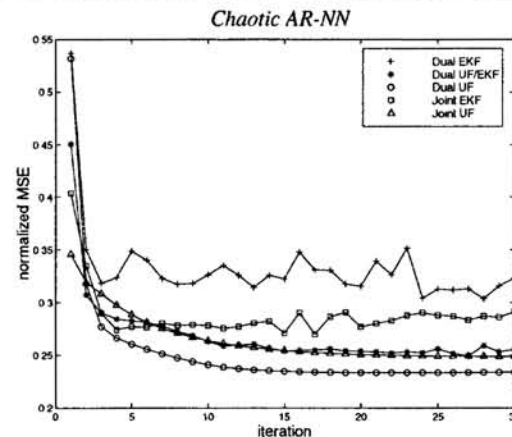

*Chaotic AR-NN*

The final table below compares *smoother* performance used for the E-step in the EM algorithm. In this case, the network models are trained on the clean time-series, and then tested on the noisy data using either the standard Kalman smoother with linearized backward model (EKS1), a Kalman smoother with a second nonlinear backward model (EKS2), and the unscented smoother (US). The forward (F), backward (B), and smoothed (S) estimation errors are reported. Again the performance benefits of the unscented approach is clear.

| Mackey-Glass | Norm. MSE | | |
|---|---|---|---|
| **Algorithm** | **F** | **B** | **S** |
| EKS1 | 0.20 | 0.70 | 0.27 |
| EKS2 | 0.20 | 0.31 | 0.19 |
| US | 0.10 | 0.24 | 0.08 |

| Chaotic AR-NN | Norm. MSE | | |
|---|---|---|---|
| **Algorithm** | **F** | **B** | **S** |
| EKS1 | 0.35 | 0.32 | 0.28 |
| EKS2 | 0.35 | 0.22 | 0.23 |
| US | 0.23 | 0.21 | 0.16 |

## 5  UF Neural Network Training

As part of the dual UF algorithm, we introduced the use of the UF for weight estimation. The approach can also be seen as a new method for the general problem of training neural networks (*i.e.*, for regression or classification problems where the input **x** is observed and

no state-estimation is required). The advantage of the UF over the EKF in this case is not as obvious, as the state-transition function is linear (See Equation 11). However, as pointed out earlier, the observation is nonlinear. Effectively, the EKF builds up an approximation to the expected Hessian by taking outer products of the gradient. The UF, however, may provide a more accurate estimate through direct approximation of the expectation of the Hessian. We have performed a number of preliminary experiments on standard benchmark data. The figure below shows the mean and std. of learning curves (computed over 100 experiments with different initial weights) for the Mackay Robot Arm Mapping dataset. Note the faster convergence, lower variance, and lower final MSE performance of the UF weight training. While these results are encouraging, further study is still necessary to fully contrast differences between UF and EKF weight training.

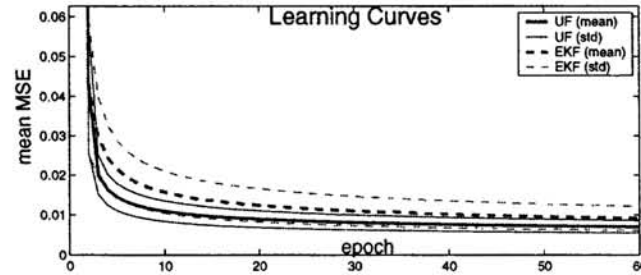

## 6 Conclusions

The EKF has been widely accepted as a standard tool in the machine learning community. In this paper we have presented an alternative to the EKF using the unscented filter. The UF consistently achieves a better level of accuracy than the EKF at a comparable level of complexity. We demonstrated this performance gain on a number of dual estimation methods as well as standard regression modeling.

### Acknowledgements

This work was sponsored in part by the NSF under grant IRI-9712346.

## Footnotes

[1]Note that a matrix square-root using the Cholesky factorization is of order $L^3/6$. However, the covariance matrices are expressed recursively, and thus the square-root can be computed in only order $L^2$ by performing a recursive update to the Cholesky factorization.

[2] An exact M-step is possible using RBF networks [2].

## References

[1] J. F. G. de Freitas, M. Niranjan, A. H. Gee, and A. Doucet. Sequential Monte Carlo methods for optimisation of neural network models. Technical Report TR-328, Cambridge University Engineering Department, Cambridge, England, November 1998.

[2] Z. Ghahramani and S. T. Roweis. Learning nonlinear dynamical systems using an EM algorithm. In M. J. Kearns, S. A. Solla, and D. A. Cohn, editors, *Advances in Neural Information Processing Systems 11: Proceedings of the 1998 Conference*. MIT Press, 1999.

[3] S. J. Julier and J. K. Uhlmann. A New Extension of the Kalman Filter to Nonlinear Systems. In *Proc. of AeroSense: The 11th International Symposium on Aerospace/Defence Sensing, Simulation and Controls, Orlando, Florida.*, 1997.

[4] R. E. Kopp and R. J. Orford. Linear regression applied to system identification for adaptive control systems. *AIAA J.*, 1:2300–06, October 1963.

[5] M. B. Matthews and G. S. Moschytz. Neural-network nonlinear adaptive filtering using the extended Kalman filter algorithm. In *INNC*, pages 115–8, 1990.

[6] A. T. Nelson. *Nonlinear Estimation and Modeling of Noisy Time-Series by Dual Kalman Filtering Methods*. PhD thesis, Oregon Graduate Institute, 1999. In preparation.

[7] S. Singhal and L. Wu. Training multilayer perceptrons with the extended Kalman filter. In *Advances in Neural Information Processing Systems 1*, pages 133–140, San Mateo, CA, 1989. Morgan Kauffman.

[8] E. A. Wan and A. T. Nelson. Dual Kalman filtering methods for nonlinear prediction, estimation, and smoothing. In *Advances in Neural Information Processing Systems 9*, 1997.